# Sparse Kernel Principal Component Analysis

**Michael E. Tipping**
Microsoft Research
St George House, 1 Guildhall St
Cambridge CB2 3NH, U.K.
mtipping@microsoft.com

## Abstract

'Kernel' principal component analysis (PCA) is an elegant nonlinear generalisation of the popular linear data analysis method, where a kernel function implicitly defines a nonlinear transformation into a feature space wherein standard PCA is performed. Unfortunately, the technique is not 'sparse', since the components thus obtained are expressed in terms of kernels associated with every training vector. This paper shows that by approximating the covariance matrix in feature space by a reduced number of example vectors, using a maximum-likelihood approach, we may obtain a highly sparse form of kernel PCA without loss of effectiveness.

## 1 Introduction

Principal component analysis (PCA) is a well-established technique for dimensionality reduction, and examples of its many applications include data compression, image processing, visualisation, exploratory data analysis, pattern recognition and time series prediction. Given a set of $N$ $d$-dimensional data vectors $\mathbf{x}_n$, which we take to have zero mean, the principal components are the linear projections onto the 'principal axes', defined as the leading eigenvectors of the sample covariance matrix $\mathbf{S} = N^{-1} \sum_{n=1}^{N} \mathbf{x}_n \mathbf{x}_n^{\mathrm{T}} = N^{-1} \mathbf{X}^{\mathrm{T}} \mathbf{X}$, where $\mathbf{X} = (\mathbf{x}_1, \mathbf{x}_2, \ldots, \mathbf{x}_N)^{\mathrm{T}}$ is the conventionally-defined 'design' matrix. These projections are of interest as they retain maximum variance and minimise error of subsequent linear reconstruction.

However, because PCA only defines a linear projection of the data, the scope of its application is necessarily somewhat limited. This has naturally motivated various developments of nonlinear 'principal component analysis' in an effort to model non-trivial data structures more faithfully, and a particularly interesting recent innovation has been 'kernel PCA' [4].

Kernel PCA, summarised in Section 2, makes use of the 'kernel trick', so effectively exploited by the 'support vector machine', in that a kernel function $k(\cdot, \cdot)$ may be considered to represent a dot (inner) product in some transformed space if it satisfies Mercer's condition — *i.e.* if it is the continuous symmetric kernel of a *positive* integral operator. This can be an elegant way to 'non-linearise' linear

procedures which depend only on inner products of the examples.

Applications utilising kernel PCA are emerging [2], but in practice the approach suffers from one important disadvantage in that it is not a *sparse* method. Computation of principal component projections for a given input $\mathbf{x}$ requires evaluation of the kernel function $k(\mathbf{x}, \mathbf{x}_n)$ in respect of all $N$ 'training' examples $\mathbf{x}_n$. This is an unfortunate limitation as in practice, to obtain the best model, we would like to estimate the kernel principal components from as much data as possible.

Here we tackle this problem by first approximating the covariance matrix in feature space by a subset of outer products of feature vectors, using a maximum-likelihood criterion based on a 'probabilistic PCA' model detailed in Section 3. Subsequently applying (kernel) PCA defines sparse projections. Importantly, the approximation we adopt is principled and controllable, and is related to the choice of the number of components to 'discard' in the conventional approach. We demonstrate its efficacy in Section 4 and illustrate how it can offer similar performance to a full non-sparse kernel PCA implementation while offering much reduced computational overheads.

## 2 Kernel PCA

Although PCA is conventionally defined (as above) in terms of the covariance, or outer-product, matrix, it is well-established that the eigenvectors of $\mathbf{X}^\mathsf{T}\mathbf{X}$ can be obtained from those of the inner-product matrix $\mathbf{X}\mathbf{X}^\mathsf{T}$. If $\mathbf{U}$ is an orthogonal matrix of column eigenvectors of $\mathbf{X}\mathbf{X}^\mathsf{T}$ with corresponding eigenvalues in the diagonal matrix $\boldsymbol{\Lambda}$, then by definition $(\mathbf{X}\mathbf{X}^\mathsf{T})\mathbf{U} = \mathbf{U}\boldsymbol{\Lambda}$. Pre-multiplying by $\mathbf{X}^\mathsf{T}$ gives:

$$(\mathbf{X}^\mathsf{T}\mathbf{X})(\mathbf{X}^\mathsf{T}\mathbf{U}) = (\mathbf{X}^\mathsf{T}\mathbf{U})\boldsymbol{\Lambda}. \tag{1}$$

From inspection, it can be seen that the eigenvectors of $\mathbf{X}^\mathsf{T}\mathbf{X}$ are $\mathbf{X}^\mathsf{T}\mathbf{U}$, with eigenvalues $\boldsymbol{\Lambda}$. Note, however, that the column vectors $\mathbf{X}^\mathsf{T}\mathbf{U}$ are not normalised since for column $i$, $\mathbf{u}_i^\mathsf{T}\mathbf{X}\mathbf{X}^\mathsf{T}\mathbf{u}_i = \lambda_i\mathbf{u}_i^\mathsf{T}\mathbf{u}_i = \lambda_i$, so the correctly normalised eigenvectors of $\mathbf{X}^\mathsf{T}\mathbf{X}$, and thus the principal axes of the data, are given by $\mathbf{U}_{\mathrm{pca}} = \mathbf{X}^\mathsf{T}\mathbf{U}\boldsymbol{\Lambda}^{-\frac{1}{2}}$.

This derivation is useful if $d > N$, when the dimensionality of $\mathbf{x}$ is greater than the number of examples, but it is also fundamental for implementing kernel PCA. In kernel PCA, the data vectors $\mathbf{x}_n$ are implicitly mapped into a feature space by a set of functions $\{\phi\} : \mathbf{x}_n \to \phi(\mathbf{x}_n)$. Although the vectors $\phi_n = \phi(\mathbf{x}_n)$ in the feature space are generally not known explicitly, their inner products are defined by the kernel: $\phi_m^\mathsf{T}\phi_n = k(\mathbf{x}_m, \mathbf{x}_n)$. Defining $\boldsymbol{\Phi}$ as the (notional) design matrix in feature space, and exploiting the above inner-product PCA formulation, allows the eigenvectors of the covariance matrix in feature space[1], $\mathbf{S}_\phi = N^{-1} \sum_n \phi_n \phi_n^\mathsf{T}$, to be specified as:

$$\mathbf{U}_{\mathrm{kpca}} = \boldsymbol{\Phi}^\mathsf{T}\mathbf{U}\boldsymbol{\Lambda}^{-\frac{1}{2}}, \tag{2}$$

where $\mathbf{U}, \boldsymbol{\Lambda}$ are the eigenvectors/values of the kernel matrix $\mathbf{K}$, with $(\mathbf{K})_{mn} = k(\mathbf{x}_m, \mathbf{x}_n)$. Although we can't compute $\mathbf{U}_{\mathrm{kpca}}$ since we don't know $\boldsymbol{\Phi}$ explicitly, we can compute *projections* of arbitrary test vectors $\mathbf{x}_* \to \phi_*$ onto $\mathbf{U}_{\mathrm{kpca}}$ in feature space:

$$\phi_*^\mathsf{T}\mathbf{U}_{\mathrm{kpca}} = \phi_*^\mathsf{T}\boldsymbol{\Phi}^\mathsf{T}\mathbf{U}\boldsymbol{\Lambda}^{-\frac{1}{2}} = \mathbf{k}_*^\mathsf{T}\mathbf{U}\boldsymbol{\Lambda}^{-\frac{1}{2}}, \tag{3}$$

where $\mathbf{k}_*$ is the $N$-vector of inner products of $\mathbf{x}_*$ with the data in kernel space: $(\mathbf{k})_n = k(\mathbf{x}_*, \mathbf{x}_n)$. We can thus compute, and plot, these projections — Figure 1 gives an example for some synthetic 3-cluster data in two dimensions.

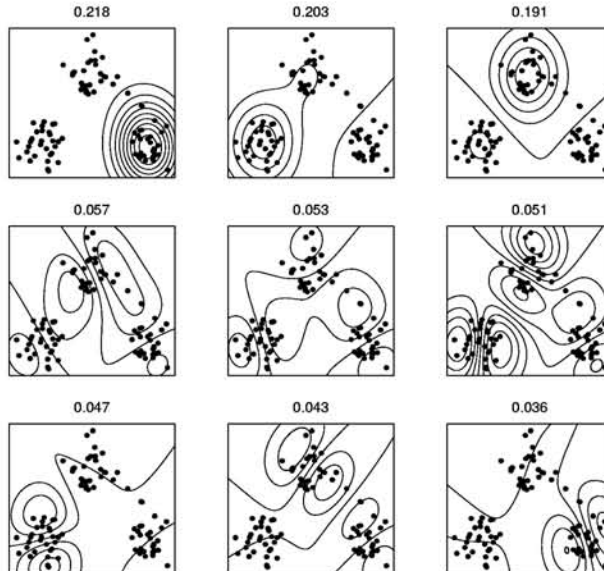

Figure 1: Contour plots of the first nine principal component projections evaluated over a region of input space for data from 3 Gaussian clusters (standard deviation 0.1; axis scales are shown in Figure 3) each comprising 30 vectors. A Gaussian kernel, $\exp(-\|\mathbf{x}-\mathbf{x}'\|^2/r^2)$, with width $r = 0.25$, was used. The corresponding eigenvalues are given above each projection. Note how the first three components 'pick out' the individual clusters [4].

## 3   Probabilistic Feature-Space PCA

Our approach to sparsifying kernel PCA is to *a priori* approximate the feature space sample covariance matrix $\mathbf{S}_\phi$ with a sum of weighted outer products of a *reduced* number of feature vectors. (The basis of this technique is thus general and its application not necessarily limited to kernel PCA.) This is achieved probabilistically, by maximising the likelihood of the feature vectors under a Gaussian density model $\phi \sim \mathcal{N}(\mathbf{0}, \mathbf{C})$, where we specify the covariance $\mathbf{C}$ by:

$$\mathbf{C} = \sigma^2 \mathbf{I} + \sum_{i=1}^{N} w_i \phi_i \phi_i^{\mathrm{T}} = \sigma^2 \mathbf{I} + \mathbf{\Phi}^{\mathrm{T}} \mathbf{W} \mathbf{\Phi}, \tag{4}$$

where $w_1 \ldots w_N$ are the adjustable weights, $\mathbf{W}$ is a matrix with those weights on the diagonal, and $\sigma^2$ is an isotropic 'noise' component common to all dimensions of feature space. Of course, a naive maximum of the likelihood under this model is obtained with $\sigma^2 = 0$ and all $w_i = 1/N$. However, if we fix $\sigma^2$, and optimise only the weighting factors $w_i$, we will find that the maximum-likelihood estimates of many $w_i$ are zero, thus realising a sparse representation of the covariance matrix.

This probabilistic approach is motivated by the fact that if we relax the form of the model, by defining it in terms of outer products of $N$ *arbitrary* vectors $\mathbf{v}_i$ (rather than the fixed training vectors), *i.e.* $\mathbf{C} = \sigma^2 \mathbf{I} + \sum_{i=1}^{N} w_i \mathbf{v}_i \mathbf{v}_i^{\mathrm{T}}$, then we realise a form of 'probabilistic PCA' [6]. That is, if $\{\mathbf{u}_i, \lambda_i\}$ are the set of eigenvectors/values of $\mathbf{S}_\phi$, then the likelihood under this model is maximised by $\mathbf{v}_i = \mathbf{u}_i$ and $w_i = (\lambda_i - \sigma^2)^{1/2}$, for those $i$ for which $\lambda_i > \sigma^2$. For $\lambda_i \leq \sigma^2$, the most likely weights $w_i$ are zero.

### 3.1 Computations in feature space

We wish to maximise the likelihood under a Gaussian model with covariance given by (4). Ignoring terms independent of the weighting parameters, its log is given by:

$$\mathcal{L} = -\frac{1}{2} \left[ N \log |\mathbf{C}| + \operatorname{tr} \left( \mathbf{C}^{-1} \mathbf{\Phi}^{\mathsf{T}} \mathbf{\Phi} \right) \right]. \tag{5}$$

Computing (5) requires the quantities $|\mathbf{C}|$ and $\phi^{\mathsf{T}} \mathbf{C}^{-1} \phi$, which for infinite dimensionality feature spaces might appear problematic. However, by judicious re-writing of the terms of interest, we are able to both compute the log-likelihood (to within a constant) and optimise it with respect to the weights. First, we can write:

$$\log |\sigma^2 \mathbf{I} + \mathbf{\Phi}^{\mathsf{T}} \mathbf{W} \mathbf{\Phi}| = D \log \sigma^2 + \log |\mathbf{W}^{-1} + \sigma^{-2} \mathbf{\Phi} \mathbf{\Phi}^{\mathsf{T}}| + \log |\mathbf{W}|. \tag{6}$$

The potential problem of infinite dimensionality, $D$, of the feature space now enters only in the first term, which is constant if $\sigma^2$ is fixed and so does not affect maximisation. The term in $|\mathbf{W}|$ is straightforward and the remaining term can be expressed in terms of the inner-product (kernel) matrix:

$$\mathbf{W}^{-1} + \sigma^{-2} \mathbf{\Phi} \mathbf{\Phi}^{\mathsf{T}} = \mathbf{W}^{-1} + \sigma^{-2} \mathbf{K}, \tag{7}$$

where $\mathbf{K}$ is the kernel matrix such that $(\mathbf{K})_{mn} = k(\mathbf{x}_m, \mathbf{x}_n)$.

For the data-dependent term in the likelihood, we can use the Woodbury matrix inversion identity to compute the quantities $\phi_n^{\mathsf{T}} \mathbf{C}^{-1} \phi_n$:

$$\phi_n^{\mathsf{T}} (\sigma^2 \mathbf{I} + \mathbf{\Phi} \mathbf{W} \mathbf{\Phi}^{\mathsf{T}})^{-1} \phi_n = \phi_n^{\mathsf{T}} \left[ \sigma^{-2} \mathbf{I} - \sigma^{-4} \mathbf{\Phi} (\mathbf{W}^{-1} + \sigma^{-2} \mathbf{\Phi}^{\mathsf{T}} \mathbf{\Phi})^{-1} \mathbf{\Phi}^{\mathsf{T}} \right] \phi_n,$$
$$= \sigma^{-2} k(\mathbf{x}_n, \mathbf{x}_n) - \sigma^{-4} \mathbf{k}_n^{\mathsf{T}} (\mathbf{W}^{-1} + \sigma^{-2} \mathbf{K})^{-1} \mathbf{k}_n, \tag{8}$$

with $\mathbf{k}_n = [k(\mathbf{x}_n, \mathbf{x}_1), k(\mathbf{x}_n, \mathbf{x}_2), \dots, k(\mathbf{x}_n, \mathbf{x}_N)]^{\mathsf{T}}$.

### 3.2 Optimising the weights

To maximise the log-likelihood with respect to the $w_i$, differentiating (5) gives us:

$$\frac{\partial \mathcal{L}}{\partial w_i} = \frac{1}{2} \left( \phi_i^{\mathsf{T}} \mathbf{C}^{-1} \mathbf{\Phi}^{\mathsf{T}} \mathbf{\Phi} \mathbf{C}^{-1} \phi_i - N \phi_i^{\mathsf{T}} \mathbf{C}^{-1} \phi_i \right), \tag{9}$$

$$= \frac{1}{2 w_i^2} \left( \sum_{n=1}^{N} \mu_{ni}^2 + N \Sigma_{ii} - N w_i \right), \tag{10}$$

where $\mathbf{\Sigma}$ and $\mu_n$ are defined respectively by

$$\mathbf{\Sigma} = (\mathbf{W}^{-1} + \sigma^{-2} \mathbf{K})^{-1}, \tag{11}$$

$$\mu_n = \sigma^{-2} \mathbf{\Sigma} \mathbf{k}_n. \tag{12}$$

Setting (10) to zero gives re-estimation equations for the weights:

$$w_i^{\text{new}} = N^{-1} \sum_{n=1}^{N} \mu_{ni}^2 + \Sigma_{ii}. \tag{13}$$

The re-estimates (13) are equivalent to *expectation-maximisation* updates, which would be obtained by adopting a factor analytic perspective [3], and introducing a set of 'hidden' Gaussian explanatory variables whose conditional means and common covariance, given the feature vectors and the current values of the weights, are given by $\mu_n$ and $\mathbf{\Sigma}$ respectively (hence the notation). As such, (13) is guaranteed to increase $\mathcal{L}$ unless it is already at a maximum. However, an alternative

re-arrangement of (10), motivated by [5], leads to a re-estimation update which typically converges significantly more quickly:

$$w_i^{\text{new}} = \frac{\sum_{n=1}^N \mu_{ni}^2}{N(1 - \Sigma_{ii}/w_i)}. \tag{14}$$

Note that these $w_i$ updates (14) are defined in terms of the computable (*i.e.* not dependent on explicit feature space vectors) quantities $\boldsymbol{\Sigma}$ and $\boldsymbol{\mu}_n$.

### 3.3 Principal component analysis

*The principal axes*

Sparse kernel PCA proceeds by finding the principal axes of the covariance model $\mathbf{C} = \sigma^2 \mathbf{I} + \boldsymbol{\Phi}^{\text{T}} \mathbf{W} \boldsymbol{\Phi}$. These are identical to those of $\boldsymbol{\Phi}^{\text{T}} \mathbf{W} \boldsymbol{\Phi}$, but with eigenvalues all $\sigma^2$ larger. Letting $\widetilde{\boldsymbol{\Phi}} = \mathbf{W}^{\frac{1}{2}} \boldsymbol{\Phi}$, then, we need the eigenvectors of $\widetilde{\boldsymbol{\Phi}}^{\text{T}} \widetilde{\boldsymbol{\Phi}}$.

Using the technique of Section 2, if the eigenvectors of $\widetilde{\boldsymbol{\Phi}} \widetilde{\boldsymbol{\Phi}}^{\text{T}} = \mathbf{W}^{\frac{1}{2}} \boldsymbol{\Phi} \boldsymbol{\Phi}^{\text{T}} \mathbf{W}^{\frac{1}{2}} = \mathbf{W}^{\frac{1}{2}} \mathbf{K} \mathbf{W}^{\frac{1}{2}}$ are $\widetilde{\mathbf{U}}$, with corresponding eigenvalues $\widetilde{\boldsymbol{\Lambda}}$, then the eigevectors/values $\{\mathbf{U}, \boldsymbol{\Lambda}\}$ of $\mathbf{C}$ that we desire are given by:

$$\mathbf{U} = \boldsymbol{\Phi}^{\text{T}} \mathbf{W}^{\frac{1}{2}} \widetilde{\mathbf{U}} \widetilde{\boldsymbol{\Lambda}}^{-\frac{1}{2}}, \tag{15}$$

$$\boldsymbol{\Lambda} = \widetilde{\boldsymbol{\Lambda}} + \sigma^2 \mathbf{I}. \tag{16}$$

*Computing projections*

Again, we can't compute the eigenvectors $\mathbf{U}$ explicitly in (15), but we can compute the projections of a general feature vector $\boldsymbol{\phi}_*$ onto the principal axes:

$$\boldsymbol{\phi}_*^{\text{T}} \mathbf{U} = \boldsymbol{\phi}_*^{\text{T}} \boldsymbol{\Phi}^{\text{T}} \mathbf{W}^{\frac{1}{2}} \widetilde{\mathbf{U}} \widetilde{\boldsymbol{\Lambda}}^{-\frac{1}{2}} = \widehat{\mathbf{k}}_*^{\text{T}} \widehat{\mathbf{P}}, \tag{17}$$

where $\widehat{\mathbf{k}}_*$ is the sparse vector containing the non-zero weighted elements of $\mathbf{k}_*$, defined earlier. The corresponding rows of $\mathbf{W}^{\frac{1}{2}} \widetilde{\mathbf{U}} \widetilde{\boldsymbol{\Lambda}}^{-\frac{1}{2}}$ are combined into a single projecting matrix $\widehat{\mathbf{P}}$, each column of which gives the coefficients of the kernel functions for the evaluation of each principal component.

### 3.4 Computing Reconstruction Error

The squared reconstruction error in kernel space for a test vector $\boldsymbol{\phi}_*$ is given by:

$$\|(\mathbf{I} - \mathbf{U}\mathbf{U}^{\text{T}})\boldsymbol{\phi}_*\|^2 = k(\mathbf{x}_*, \mathbf{x}_*) - \widehat{\mathbf{k}}_*^{\text{T}} \widehat{\mathbf{K}}^{-1} \widehat{\mathbf{k}}_*, \tag{18}$$

with $\widehat{\mathbf{K}}$ the kernel matrix evaluated only for the representing vectors.

## 4 Examples

To obtain sparse kernel PCA projections, we first specify the noise variance $\sigma^2$, which is the the amount of variance per co-ordinate that we are prepared to allow to be explained by the (structure-free) isotropic noise rather than with the principal axes (this choice is a surrogate for deciding how many principal axes to retain in conventional kernel PCA). Unfortunately, the measure is in feature space, which makes it rather more difficult to interpret than if it were in data space (equally so, of course, for interpretation of the eigenvalue spectrum in the non-sparse case).

We apply sparse kernel PCA to the Gaussian data of Figure 1 earlier, with the same kernel function and specifying $\sigma = 0.25$, deliberately chosen to give nine representing kernels so as to facilitate comparison. Figure 2 shows the nine principal component projections based on the approximated covariance matrix, and gives qualitatively equivalent results to Figure 1 while utilising only 10% of the kernels. Figure 3 shows the data and highlights those examples corresponding to the nine kernels with non-zero weights. Note, although we do not consider this aspect further here, that these representing vectors are themselves highly informative of the structure of the data (*i.e.* with a Gaussian kernel, for example, they tend to represent distinguishable clusters). Also in Figure 3, contours of reconstruction error, based only on those nine kernels, are plotted and indicate that the nonlinear model has more faithfully captured the structure of the data than would standard linear PCA.

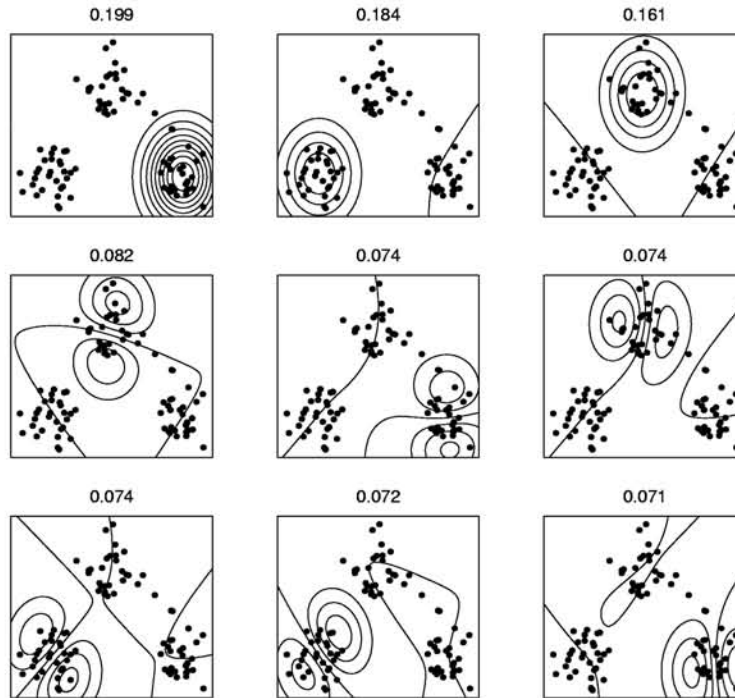

Figure 2: The nine principal component projections obtained by sparse kernel PCA.

To further illustrate the fidelity of the sparse approximation, we analyse the 200 training examples of the 7-dimensional 'Pima Indians diabetes' database [1]. Figure 4 (left) shows a plot of reconstruction error against the number of principal components utilised by both conventional kernel PCA and its sparse counterpart, with $\sigma^2$ chosen so as to utilise 20% of the kernels (40). An expected small reduction in accuracy is evident in the sparse case. Figure 4 (right) shows the error on the associated test set when using a linear support vector machine to classify the data based on those numbers of principal components. Here the sparse projections actually perform marginally better on average, a consequence of both randomness and, we note with interest, presumably some inherent complexity control implied by the use of a sparse approximation.

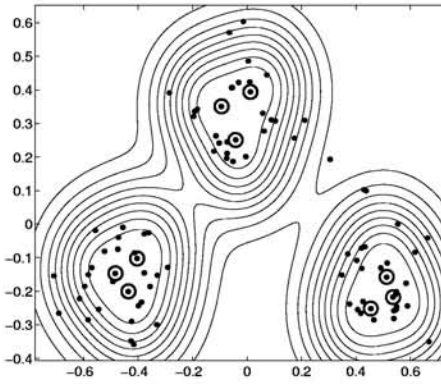

Figure 3: The data with the nine representing kernels circled and contours of reconstruction error (computed in *feature space* although displayed as a function of **x**) overlaid.

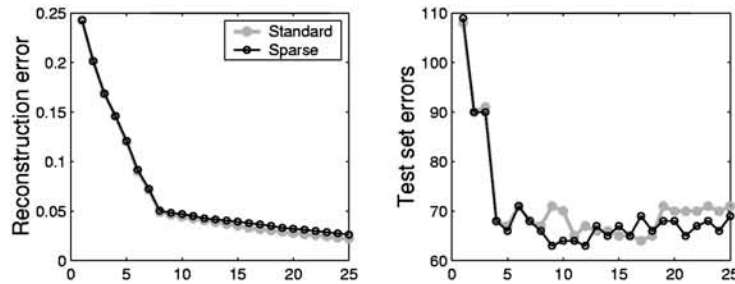

Figure 4: RMS reconstruction error (left) and test set misclassifications (right) for numbers of retained principal components ranging from 1–25. For the standard case, this was based on all 200 training examples, for the sparse form, a subset of 40. A Gaussian kernel of width 10 was utilised, which gives near-optimal results if used in an SVM classification.

## Footnotes

[1]Here, and in the rest of the paper, we do not 'centre' the data in feature space, although this may be achieved if desired (see [4]). In fact, we would argue that when using a Gaussian kernel, it does not necessarily make sense to do so.

# References

[1] B. D. Ripley. *Pattern Recognition and Neural Networks.* Cambridge University Press, Cambridge, 1996.

[2] S. Romdhani, S. Gong, and A. Psarrou. A multi-view nonlinear active shape model using kernel PCA. In *Proceedings of the 1999 British Machine Vision Conference*, pages 483–492, 1999.

[3] D. B. Rubin and D. T. Thayer. EM algorithms for ML factor analysis. *Psychometrika*, 47(1):69–76, 1982.

[4] B. Schölkopf, A. Smola, and K.-R. Müller. Nonlinear component analysis as a kernel eigenvalue problem. *Neural Computation*, 10:1299–1319, 1998. Technical Report No. 44, 1996, Max Planck Institut für biologische Kybernetik, Tübingen.

[5] M. E. Tipping. The Relevance Vector Machine. In S. A. Solla, T. K. Leen, and K.-R. Müller, editors, *Advances in Neural Information Processing Systems 12*, pages 652–658. Cambridge, Mass: MIT Press, 2000.

[6] M. E. Tipping and C. M. Bishop. Probabilistic principal component analysis. *Journal of the Royal Statistical Society, Series B*, 61(3):611–622, 1999.
